# Managing Power Consumption and Performance of Computing Systems Using Reinforcement Learning

**Gerald Tesauro, Rajarshi Das, Hoi Chan, Jeffrey O. Kephart,**
**Charles Lefurgy**[*]**, David W. Levine and Freeman Rawson**[*]
IBM Watson and Austin[*] Research Laboratories
{gtesauro,rajarshi,hychan,kephart,lefurgy,dwl,frawson}@us.ibm.com

## Abstract

Electrical power management in large-scale IT systems such as commercial data-centers is an application area of rapidly growing interest from both an economic and ecological perspective, with billions of dollars and millions of metric tons of $CO_2$ emissions at stake annually. Businesses want to save power without sacrificing performance. This paper presents a reinforcement learning approach to simultaneous online management of both performance and power consumption. We apply RL in a realistic laboratory testbed using a Blade cluster and dynamically varying HTTP workload running on a commercial web applications middleware platform. We embed a CPU frequency controller in the Blade servers' firmware, and we train policies for this controller using a multi-criteria reward signal depending on both application performance and CPU power consumption. Our testbed scenario posed a number of challenges to successful use of RL, including multiple disparate reward functions, limited decision sampling rates, and pathologies arising when using multiple sensor readings as state variables. We describe innovative practical solutions to these challenges, and demonstrate clear performance improvements over both hand-designed policies as well as obvious "cookbook" RL implementations.

## 1 Introduction

Energy consumption is a major and growing concern throughout the IT industry as well as for customers and for government regulators concerned with energy and environmental matters. To cite a prominent example, the US Congress recently mandated a study of the power efficiency of servers, including a feasibility study of an Energy Star standard for servers and data centers [16]. Growing interest in power management is also apparent in the formation of the Green Grid, a consortium of systems and other vendors dedicated to improving data center power efficiency [7]. Recent trade press articles also make it clear that computer purchasers and data center operators are eager to reduce power consumption and the heat densities being experienced with current systems.

In response to these concerns, researchers are tackling intelligent power control of processors, memory chips and whole systems, using technologies such as processor throttling, frequency and voltage manipulation, low-power DRAM states, feedback control using measured power values, and packing and virtualization to reduce the number of machines that need to be powered on to run a workload.

This paper presents a reinforcement learning (RL) approach to developing effective control policies for real-time management of power consumption in application servers. Such power management policies must make intelligent tradeoffs between power and performance, as running servers in low-power modes inevitably degrades the application performance. Our approach to this entails designing a multi-criteria objective function $U_{pp}$ taking both power and performance into account, and using it to give reward signals in reinforcement learning. We let $U_{pp}$ be a function of mean

application response time $RT$, and total power Pwr consumed by the servers in a decision interval. Specifically, $U_{pp}$ subtracts a linear power cost from a performance-based utility $U(RT)$:

$$U_{pp}(RT, \text{Pwr}) = U(RT) - \epsilon * \text{Pwr} \qquad (1)$$

where $\epsilon$ is a tunable coefficient expressing the relative value of power and performance objectives. This approach admits other objective functions such as "performance value per watt" $U_{pp} = U(RT)/\text{Pwr}$, or a simple performance-based utility $U_{pp} = U(RT)$ coupled with a constraint on total power.

The problem of jointly managing performance and power in IT-systems was only recently studied in the literature [5, 6, 17]. Existing approaches use knowledge-intensive and labor-intensive modeling, such as developing queuing-theoretic or control-theoretic performance models. RL methods can potentially avoid such knowledge bottlenecks, by automatically learning high-quality management policies using little or no built-in system specific knowledge. Moreover, as we discuss later, RL may have the merit of properly handling complex dynamic and delayed consequences of decisions.

In Section 2 we give details of our laboratory testbed, while Section 3 describes our RL approach. Results are presented in Section 4, and the final section discusses next steps in our ongoing research and ties to related work.

## 2 Experimental Testbed

Figure 1 provides a high-level overview of our experimental testbed. In brief, a Workload Generator produces an HTTP-based workload of dynamically varying intensity that is routed to a *blade cluster*, i.e., a collection of blade servers contained in a single chassis. (Specifically, we use an IBM BladeCenter containing xSeries HS20 blade servers.) A commercial performance manager and our RL-based power manager strive to optimize a joint power-performance objective cooperatively as load varies, each adjusting its control parameters individually while sharing certain information with the other manager. RL techniques (described subsequently) are used to train a state-action value function which defines the power manager's control policy. The "state" is characterized by a set of observable performance, power and load intensity metrics collected in our data collection module as detailed below. The "action" is a throttling of CPU frequency[1] that is achieved by setting a "powercap" on each blade that provides an upper limit on the power that the blade may consume. Given this limit, a feedback controller embedded in the server's firmware [11] continuously monitors the power consumption, and continuously regulates the CPU clock speed so as to keep the power consumption close to, but not over, the powercap limit. The CPU throttling affects both application performance as well as power consumption, and the goal of learning is to achieve the optimal level of throttling in any given state that maximizes cumulative discounted values of joint reward $U_{pp}$.

We control workload intensity by varying the number of clients $n_c$ sending HTTP requests. We varied $n_c$ in a range from 1 to 50 using a statistical time-series model of web traffic derived from observations of a highly accessed Olympics web site [14]. Clients behave according to a closed-loop model [12] with exponentially distributed think times of mean 125 msec.

The commercial performance manager is WebSphere Extended Deployment (WXD)[18], a multi-node webserver environment providing extensive data collection and performance management functionality. WXD manages the routing policy of the Workload Distributer as well as control parameters on individual blades, such as the maximum workload concurrency.

Our data collector receives several streams of data and provides a synchronized report to the power policy evaluator on a time scale $\tau_l$ (typically set to 5 seconds). Data generated on much faster time scales than $\tau_l$ are time-averaged over the interval, otherwise the most recent values are reported. Among the aggregated data are several dozen performance metrics collected by a daemon running on the WXD data server, such as mean response time, queue length and number of CPU cycles per transaction; CPU utilization and effective frequency collected by local daemons on each blade; and current power and temperature measurements collected by the firmware on each blade, which are polled using IPMI commands sent from the BladeCenter management module.

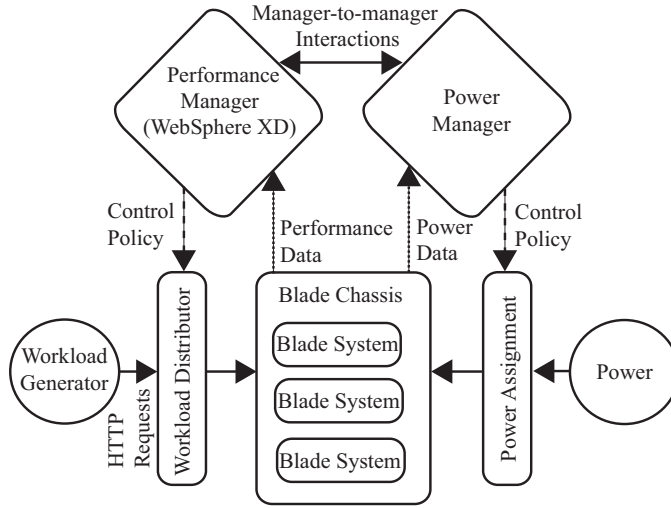

Figure 1: Overview of testbed environment.

## 2.1 Utility function definition

Our specific performance-based utility $U(RT)$ in Eq. 1 is a piecewise linear function of response time $RT$ which returns a maximum value of 1.0 when $RT$ is less than a specified threshold $RT_0$, and which drops linearly when $RT$ exceeds $RT_0$, i.e.,

$$U(RT/RT_0) = \begin{cases} 1.0 & \text{if } RT \leq RT_0 \\ 2.0 - RT/RT_0 & \text{otherwise} \end{cases} \qquad (2)$$

Such a utility function reflects the common assumptions in customer service level agreements that there is no incentive to improve the performance once it reaches the target threshold, and that there is always a constant incentive to improve performance if it violates the threshold. In all of our experiments, we set $RT_0$ = 1000 msec, and we also set the power scale factor $\epsilon = 0.01$ in Eq. 1. At this value of $\epsilon$ the power-performance tradeoff is strongly biased in favor of performance, as is commonly desired in today's data centers. However, larger values of $\epsilon$ could be appropriate in future scenarios where power is much more costly, in which case the optimal policies would tolerate more frequent performance threshold violations in order to save more aggressively on power consumption.

## 2.2 Baseline Powercap Policies

To assess the effectiveness of our RL-based power management policies, we compare with two different benchmark policies: "UN" (unmanaged) and "HC" (hand-crafted). The unmanaged policy always sets the powercap to a maximal value of 120W; we verified that the CPU runs at the highest frequency under all load conditions with this setting.

The hand-crafted policy was created as follows. We measured power consumption on a blade server at extremely low ($n_c = 1$) and high ($n_c = 50$) loads, finding that in all cases the power consumption ranged between 75 and 120 watts. Given this range, we established a grid of sample points, with $p_\kappa$ running from 75 watts to 120 watts in increments of 5 watts, and the number of clients running from 0 to 50 in increments of 5. For each of the 10 possible settings of $p_\kappa$, we held $n_c$ fixed at 50 for 45 minutes to permit WXD to adapt to the workload, and then decremented $n_c$ by 5 every 5 minutes. Finally, the models $RT(p_\kappa, n_c)$ and $Pwr(p_\kappa, n_c)$, were derived by linearly interpolating for the $RT$ and $Pwr$ between the sampled grid points.

We substitute these models into our utility function $U_{pp}(RT, Pwr)$ to obtain an equivalent utility function $U'$ depending on $p_\kappa$ and $n_c$, i.e., $U'(p_\kappa, n_c) = U_{pp}(RT(p_\kappa, n_c), Pwr(p_\kappa, n_c))$. We can then choose the optimal powercap for any workload intensity $n_c$ by optimizing $U'$: $p_\kappa^*(n_c) = \arg\max_{p_\kappa} U'(p_\kappa, n_c)$.

# 3   Reinforcement Learning Approach

One may naturally question whether RL could be capable of learning effective control policies for systems as complex as a population of human users interacting with a commercial web application. Such systems are surely far from full observability in the MDP sense. Without even considering whether the behavior of users is "Markovian," we note that the state of a web application may depend, for example, on the states of the underlying middleware and Java Virtual Machines (JVMs), and these states are not only unobservable, they also have complex historical dependencies on prior load and performance levels over multiple time scales. Despite such complexities, we have found in our earlier work [15, 9] that RL can in fact learn decent policies when using severely limited state descriptions, such as a single state variable representing current load intensity. The focus of our work in this paper is to examine empirically whether RL may obtain better policies by including more observable metrics in the state description.

Another important question is whether current decisions have long-range effects, or if it suffices to simply learn policies that optimize immediate reward. The answer appears to vary in an interesting way: under low load conditions, the system response to a decision is fairly immediate, whereas under conditions of high queue length (which may result from poor throttling decisions), the responsiveness to decisions may become sluggish and considerably delayed.

Our reinforcement learning approach leverages our recent "Hybrid RL" approach [15], which originally was applied to autonomic server allocation. Hybrid RL is a form of offline (batch) RL that entails devising an initial control policy, running the initial policy in the live system and logging a set of (state, action, reward) tuples, and then using a standard RL/function approximator combination to learn a value function $V(s, a)$ estimating cumulative expected reward of taking action $a$ in state $s$. (The term "Hybrid" refers to the fact that expert domain knowledge can be engineered into the initial policy without needing explicit engineering or interfacing into the RL module.) The learned value function $V$ then implies a policy of selecting the action $a^*$ in state $s$ with highest expected value, i.e., $a^* = \arg\max_a V(s, a)$.

For technical reasons detailed below, we use the Sarsa(0) update rule rather than Q-Learning (note that unlike textbook Sarsa, decisions are made by an external fixed policy). Following [15], we set the discount parameter $\gamma = 0.5$; we found some preliminary evidence that this is superior to setting $\gamma = 0.0$ but haven't been able to systematically study the effect of varying $\gamma$. We also perform standard direct gradient training of neural net weights: we train a multilayer perceptron with 12 sigmoidal hidden units, using backprop to compute the weight changes. Such an approach is appealing, as it is simple to implement and has a proven track record of success in many practical applications. There is a theoretical risk that the approach could produce value function divergence. However, we have not seen such divergence in our application. Were it to occur, it would not entail any live performance costs, since we train offline. Additionally, we note that instead of direct gradient training, we can use Baird's residual gradient method [4], which guarantees convergence to local Bellman error minima. In practice we find that direct gradient training yields good convergence to Bellman error minima in $\sim$5-10K training epochs, requiring only a few CPU minutes on a 3GHz workstation.

In implementing an initial policy to be used with Hybrid RL, one would generally want to exploit the best available human-designed policy, combined with sufficient randomized exploration needed by RL, in order to achieve the best possible learned policy. However, in view of the difficulty expected in designing such initial policies, it would be advantageous to be able to learn effective policies starting from simplistic initial policies. We have therefore trained our RL policies using an extremely simple performance-biased random walk policy for setting the powercap, which operates as follows: At every decision point, $p_\kappa$ either is increased by 1 watt with probability $p_+$, or decreased by 1 watt with probability $p_- = (1 - p_+)$. The upward bias $p_+$ depends on the ratio $r = \mathrm{RT}/\mathrm{RT}_0$ of current mean response time to response time threshold according to: $p_+ = r/(1 + r)$. Note that this rule implies an unbiased random walk when $r = 1$ and that $p_+ \to 1$ for $r \gg 1$, while $p_+ \to 0$ when $r \ll 1$. This simple rule seems to strike a good balance between keeping the performance near the desired threshold, while providing plenty of exploration needed by RL, as can been seen in Figure 2.

Having collected training data during the execution of an initial policy, the next step of Hybrid RL is to design an (input, output) representation and functional form of the value function approximator.

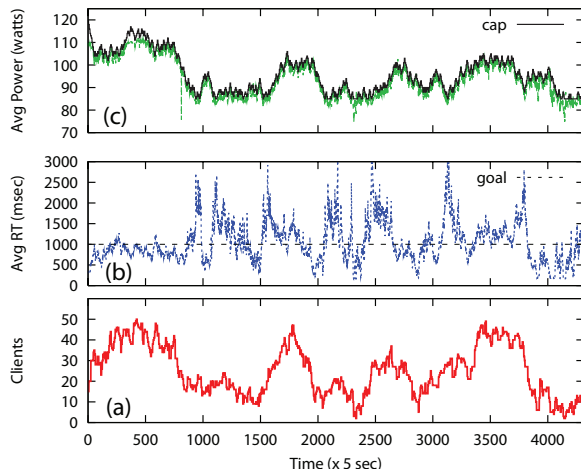

Figure 2: Traces of (a) workload intensity, (b) mean response time, and (c) powercap and consumed power of the random-walk (RW) powercap policy.

We have initially used the basic input representation studied in [15], in which the state $s$ is represented using a single metric of workload intensity (number of clients $n_c$), and the action $a$ is a single scalar variable—the powercap $p_\kappa$. This scheme robustly produces decent learned policies, with little sensitivity to exact learning algorithm parameter settings. In later experiments, we have expanded the state representation to a much larger set of 14 state variables, and find that substantial improvements in learned policies can be obtained, provided that certain data pre-processing techniques are used, as detailed below.

## 3.1   System-specific innovations

In our research in this application domain, we have devised several innovative "tricks" enabling us to achieve substantially improved RL performance. Such tricks are worth mentioning as they are likely to be of more general use in other problem domains with similar characteristics.

First, to represent and learn $V$, we could employ a single output unit, trained on the total utility (reward) using Q-Learning. However, we can take advantage of the fact that total utility $U_{pp}$ in equation 1 is a linear combination of performance utility $U$ and power cost $-\epsilon * Pwr$. Since the separate reward components are generally observable, and since these should have completely different functional forms relying on different state variables, we propose training two separate function approximators estimating future discounted reward components $V_{perf}$ and $V_{pwr}$ respectively. This type of "decompositional reward" problem has been studied for tabular RL in [13], where it is shown that learning the value function components using Sarsa provably converges to the correct total value function. (Note that Q-Learning cannot be used to train the value function components, as it incorrectly assumes that the optimal policy optimizes each individual component function.)

Second, we devised a new type of neuronal output unit to learn $V_{perf}$. This is motivated by the shape of $U$, which is a piecewise linear function of $RT$, with constant value for low $RT$ and linearly decreasing for large $RT$. This functional form is is not naturally approximated by either a linear or a sigmoidal transfer function. However, by noting that the derivative of $U$ is a step function (changing from 0 to -1 at the threshold), and that sigmoids give a good approximation to step functions, this suggests using an output transfer function that behaves as the integral of a sigmoid function. Specifically, our transfer function has the form $Y(x) = 1 - \chi(x)$ where $\chi(x) = \int \sigma(x)dx + C$, where $\sigma(x) = 1/(1 + \exp(-x))$ is the standard sigmoid function, and the integration constant $C$ is chosen so that $\chi \to 0$ as $x \to -\infty$. We find that this type of output unit is easily trained by standard backprop and provides quite a good approximation to the true expected rewards.

We have also trained separate neural networks to estimate $V_{pwr}$ using a similar hidden layer architecture and a standard linear output unit. However, we found only a slight improvement in Bellman error over a simple estimator of predicted power $\cong p_\kappa$ (although this is not always a good estimate).

Hence for simplicity we used $V_{pwr} = -\epsilon * p_\kappa$ in computing the overall learned policy maximizing $V = V_{perf} + V_{pwr}$.

Thirdly, we devised a data pre-processing technique to address a specific rate limitation in our system that the powercap decision $p_\kappa$ as well as the number of clients $n_c$ can only be changed every 30 seconds, whereas we collect state data from the system every 5 seconds. This limitation was imposed because faster variations in effective CPU speed or in load disrupt WXD's functionality, as its internal models estimate parameters on much slower time scales, and in particular, it assumes that CPU speed is a constant. As a result, we cannot do standard RL on the 5 second interval data, since this would presume the policy's ability to make a new decision every 5 seconds. A simple way to address this would be to discard data points where a decision was not made (5/6 of the data), but this would make the training set much smaller, and we would lose valuable state transition information contained in the discarded samples. As an alternative, we divide the entire training set into six subsets according to line number mod-6, so that within each subset, adjacent data points are separated by 30 second intervals. We then concatenate the subsets to form one large training set, with no loss of data, where all adjacent intervals are 30 seconds long. In effect, a sweep through such a dataset replays the experiment six times, corresponding to the six different 5-second phases within the 30-second decision cycle. As we shall see in the following section, such rearranged datasets result in substantially more stable policies.

Finally, we realized that in the artificially constructed dataset described above, there is an inaccuracy in training on samples in the five non-decision phases: standard RL would presume that the powercap decision is held constant over the full 30 seconds until the next recorded sample, whereas we know that decision actually changes somewhere in the middle of the interval, depending on the phase. To obtain the best approximation to a constant decision over such intervals, we compute an equally weighted average $\bar{p}_\kappa$ of the recorded decisions at times $\{t, t+5, t+10, t+15, t+20, t+25\}$ and train on $\bar{p}_\kappa$ as the effective decision that was made at time $t$. This change results in a significant reduction ($\sim 40\%$) in Bellman error, and the combination of this with the mod-6 data reordering enables us to obtain substanial improvements in policy performance.

## 4 Results

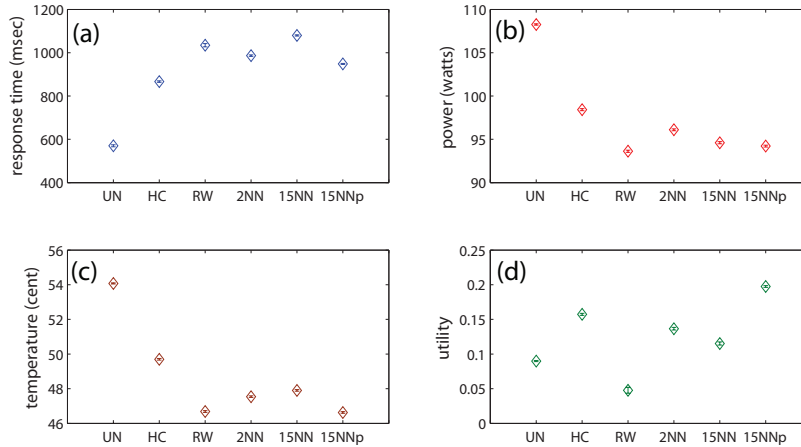

Figure 3: Comparison of mean metrics (a) response time, (b) power consumed, (c) temperature and (d) utility for six different power management policies: "UN" (unmanaged), "HC" (hand-crafted), "RW" (random walk), "2NN" (2-input neural net), "15NN" (15-input neural net, no pre-processing), "15NNp" (15-input neural net with pre-processing).

While we have conducted experiments in other work involving multiple blade servers, in this section we focus on experiments involving a single blade. Fig. 3 plots various mean performance metrics in identical six-hour test runs using identical workload traces for six different power management policies: "UN" and "HC" denote the unmanaged and hand-crafted policies described in Sec. 2.2; "RW" is the random-walk policy of Sec. 3; "2NN" denotes a two-input (single state variable) neural net; "15NN" refers to a 15-input neural net without any data pre-processing as described in Sec. 3.1, and

"15NNp" indicates a 15-input neural net using said pre-processing. In the figure, the performance metrics plotted are: (a) mean response time, (b) mean power consumed, (c) mean temperature, and most importantly, (d) mean utility. Standard error in estimates of these mean values are quite small, as indicated by error bars which lie well within the diamond-shaped data points. Since the runs use identical workload traces, we can also assess significance of the differences in means across policies via paired T-tests; exhaustive pairwise comparisons show that in all cases, the null hypothesis of no difference in mean metrics is rejected at 1% significance level with P-value $\leq 10^{-6}$.

We see in Fig. 3 that all RL-based policies, after what is effectively a single round of policy iteration, significantly outperform the original random walk policy which generated the training data. Using only load intensity as a state variable, 2NN achieves utility close to (but not matching) the hand-crafted policy. 15NN is disappointing in that its utility is actually worse than 2NN, for reasons that we discuss below. Comparing 15NNp with 15NN shows that pre-processing yields great improvements; 15NNp is clearly the best of the six policies. Breaking down overall utility into separate power and performance components, we note that all RL-based policies achieve greater power savings than HC at the price of somewhat higher mean response times. An additional side benefit of this is lower mean temperatures, as shown in the lower left plot; this implies both lower cooling costs as well as prolonged machine life.

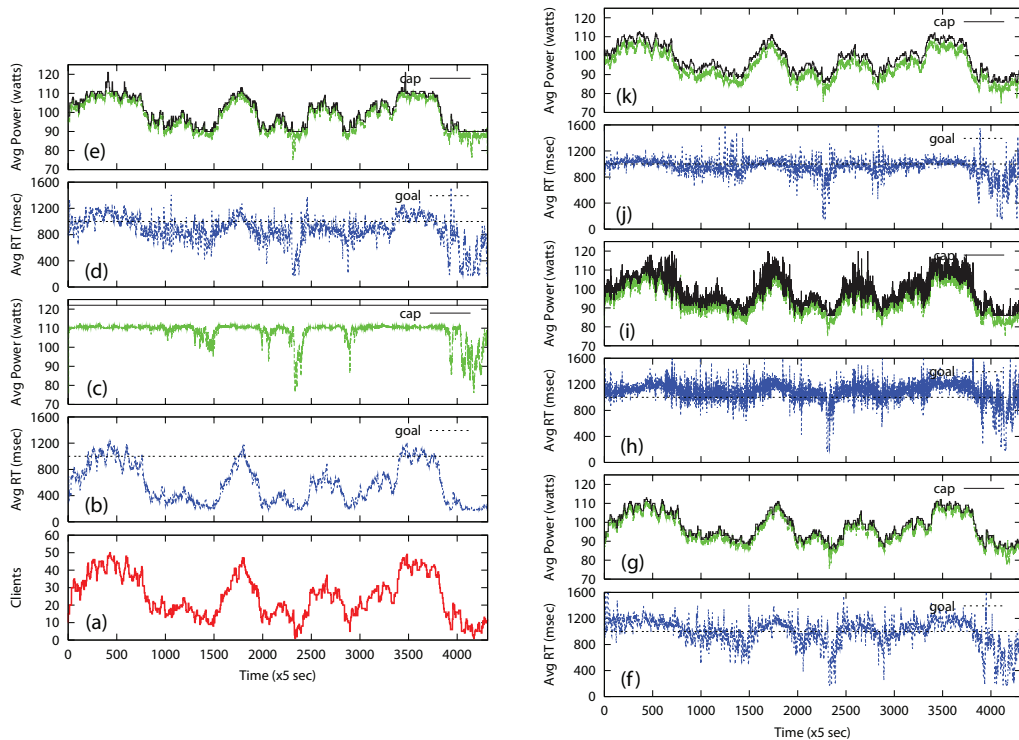

Figure 4: Traces of the five non-random policies: (a) workload intensity; (b) UN response time; (c) UN powercap; (d) HC response time; (e) HC powercap; (f) 2NN response time; (g) 2NN powercap; (h) 15NN response time; (i) 15NN powercap; (j) 15NNp response time; (k) 15NNp powercap.

Fig. 4 shows the actual traces of response time, powercap and power consumed in all experiments except the random walk, which was plotted earlier. The most salient points to note are that 15NNp exhibits the steadiest response time, keeping closest to the response time goal, and that the powercap decsions of 15NN show quite large short-term fluctuations. We attribute the latter behavior to "overreacting" to response time fluctuations above or below the target value. Such behavior may well be correct if the policy could reset every 5 seconds, as 15NN presumes. In this case, the policy could react to a response time flucutation by setting an extreme powercap value in an attempt to quickly drive the response time back to the goal value, and then backing off to a less extreme value 5 seconds later. However, such behavior would be quite poor in the actual system, in which the extreme powercap setting is held fixed for 30 seconds.

## 5 Summary and related work

This paper presented a successful application of batch RL combined with nonlinear function approximation in a new and challenging domain of autonomic management of power and performance in web application servers. We addressed challenges arising both from operating in real hardware, and from limitations imposed by interoperating with commercial middleware. By training on data from a simple random-walk initial policy, we achieved high-quality management polices that outperformed the best available hand-crafted policy. Such policies save more than 10% on server power while keeping performance close to a desired target.

In our ongoing and future work, we are aiming to scale the approach to an entire Blade cluster, and to achieve much greater levels of power savings. With the existing approach it appears that power savings closer to 20% could be obtained simply by using more realistic web workload profiles in which high-intensity spikes are brief, and the ratio of peak-to-mean workload is much higher than in our current traffic model. It also appears that savings of $\sim$30% are plausible when using multi-core processors [8]. Finally, we are also aiming to learn policies for powering machines off when feasible; this offers the potential to achieve power savings of 50% or more. In order to scale our approach to larger systems, we can leverage the fact that Blade clusters usually have sets of identical machines. All servers within such a homogeneous set can be managed in an identical fashion by the performance and power managers, thereby making the size of the overall state space and the action space more tractable for RL.

An important component of our future work is also to improve our current RL methodology. Beyond Hybrid RL, there has been much recent research in offline RL methods, including LSPI [10], Apprenticeship Learning [2], Differential Dynamic Programming [1], and fitted policy iteration minimizing Bellman residuals [3]. These methods are of great interest to us, as they typically have stronger theoretical guarantees than Hybrid RL, and have delivered impressive performance in applications such as helicopter aerobatics. For powering machines on and off, we are especially interested in offline model-based RL approaches: as the number of training samples that can be acquired is likely to be severely limited, it will be important to reduce sample complexity by learning explicit state-transition models.

## Footnotes

[1]An alternative technique with different power/performance trade-offs is Dynamic Voltage and Frequency Scaling (DVFS).

## References

[1] P. Abbeel, A. Coates, M. Quigley, and A. Y. Ng. An application of reinforcement learning to aerobatic helicopter flight. In *Proc. of NIPS-06*, 2006.

[2] P. Abbeel and A. Y. Ng. Exploration and apprenticeship learning in reinforcement learning. In *Proc. of ICML-05*, 2005.

[3] A. Antos, C. Szepesvari, and R. Munos. Learning near-optimal policies with bellman-residual minimization based fitted policy iteration and a single sample path. In *Proc. of COLT-06*, 2006.

[4] L. Baird. Residual algorithms: Reinforcement learning with function approximation. In *Proc. of ICML-95*, 1995.

[5] Y. Chen et al. Managing server energy and operational costs in hosting centers. In *Proc. of SIGMETRICS*, 2005.

[6] M. Femal and V. Freeh. Boosting data center performance through non-uniform power allocation. In *Second Intl. Conf. on Autonomic Computing*, 2005.

[7] Green Grid Consortium. Green grid. http://www.thegreengrid.org, 2006.

[8] J. Chen et al. Datacenter power modeling and prediction. UC Berkeley RAD Lab presentation, 2007.

[9] J. O. Kephart, H. Chan, R. Das, D. Levine, G. Tesauro, F. Rawson, and C. Lefurgy. Coordinating multiple autonomic managers to achieve specified power-performance tradeoffs. In *Proc. of ICAC-07*, 2007.

[10] M. G. Lagoudakis and R. Parr. Least-squares policy iteration. *J. of Machine Learning Research*, 4:1107–1149, 2003.

[11] C. Lefurgy, X. Wang, and M. Ware. Server-level power control. In *Proc. of ICAC-07*, 2007.

[12] D. Menasce and V. A. F. Almeida. *Capacity Planning for Web Performance: Metrics, Models, and Methods*. Prentice Hall, 1998.

[13] S. Russell and A. L. Zimdars. Q-decomposition for reinforcement learning agents. In *Proc. of ICML-03*, pages 656–663, 2003.

[14] M. S. Squillante, D. D. Yao, and L. Zhang. Internet traffic: Periodicity, tail behavior and performance implications. In *System Performance Evaluation: Methodologies and Applications*, 1999.

[15] G. Tesauro, N. K. Jong, R. Das, and M. N. Bennani. A hybrid reinforcement learning approach to autonomic resource allocation. In *Proc. of ICAC-06*, pages 65–73, 2006.

[16] United States Environmental Protection Agency. Letter to Enterprise Server Manufacturers and Other Stakeholders. http://www.energystar.gov, 2006.

[17] M. Wang et al. Adaptive Performance Control of Computing Systems via Distributed Cooperative Control: Application to Power Management in Computer Clusters. In *Proc. of ICAC-06*, 2006.

[18] WebSphere Extended Deployment. http://www.ibm.com/software/webservers/appserv/extend/, 2007.

